# A Theory of Mean Field Approximation

**T. Tanaka**
Department of Electronics and Information Engineering
Tokyo Metropolitan University
1-1, Minami-Osawa, Hachioji, Tokyo 192-0397 Japan

## Abstract

I present a theory of mean field approximation based on information geometry. This theory includes in a consistent way the naive mean field approximation, as well as the TAP approach and the linear response theorem in statistical physics, giving clear information-theoretic interpretations to them.

## 1 INTRODUCTION

Many problems of neural networks, such as learning and pattern recognition, can be cast into a framework of statistical estimation problem. How difficult it is to solve a particular problem depends on a statistical model one employs in solving the problem. For Boltzmann machines[1] for example, it is computationally very hard to evaluate expectations of state variables from the model parameters.

Mean field approximation[2], which is originated in statistical physics, has been frequently used in practical situations in order to circumvent this difficulty. In the context of statistical physics several advanced theories have been known, such as the TAP approach[3], linear response theorem[4], and so on. For neural networks, application of mean field approximation has been mostly confined to that of the so-called naive mean field approximation, but there are also attempts to utilize those advanced theories[5, 6, 7, 8].

In this paper I present an information-theoretic formulation of mean field approximation. It is based on information geometry[9], which has been successfully applied to several problems in neural networks[10]. This formulation includes the naive mean field approximation as well as the advanced theories in a consistent way. I give the formulation for Boltzmann machines, but its extension to wider classes of statistical models is possible, as described elsewhere[11].

## 2 BOLTZMANN MACHINES

A Boltzmann machine is a statistical model with $N$ binary random variables $s_i \in \{-1, 1\}$, $i = 1, \ldots, N$. The vector $s = (s_1, \ldots, s_N)$ is called the state of the Boltzmann machine.

The state $s$ is also a random variable, and its probability law is given by the Boltzmann-Gibbs distribution

$$p(s) = e^{-E(s) - \psi(p)}, \tag{1}$$

where $E(s)$ is the "energy" defined by

$$E(s) = -\sum_i h^i s_i - \sum_{\langle ij \rangle} w^{ij} s_i s_j \tag{2}$$

with $h^i$ and $w^{ij}$ the parameters, and $-\psi(p)$ is determined by the normalization condition and is called the Helmholtz free energy of $p$. The notation $\langle ij \rangle$ means that the summation should be taken over all distinct pairs.

Let $\eta_i(p) \equiv \langle s_i \rangle_p$ and $\eta_{ij}(p) \equiv \langle s_i s_j \rangle_p$, where $\langle \cdot \rangle_p$ means the expectation with respect to $p$. The following problem is essential for Boltzmann machines:

**Problem 1**   Evaluate the expectations $\eta_i(p)$ and $\eta_{ij}(p)$ from the parameters $h^i$ and $w^{ij}$ of the Boltzmann machine $p$.

## 3   INFORMATION GEOMETRY

### 3.1   ORTHOGONAL DUAL FOLIATIONS

A whole set $\mathcal{M}$ of the Boltzmann-Gibbs distribution (1) realizable by a Boltzmann machine is regarded as an exponential family. Let us use shorthand notations $I, J, \ldots$, to represent distinct pairs of indices, such as $ij$. The parameters $h^i$ and $w^I$ constitute a coordinate system of $\mathcal{M}$, called the canonical parameters of $\mathcal{M}$. The expectations $\eta_i$ and $\eta_I$ constitute another coordinate system of $\mathcal{M}$, called the expectation parameters of $\mathcal{M}$.

Let $\mathcal{F}_0$ be a subset of $\mathcal{M}$ on which $w^I$ are all equal to zero. I call $\mathcal{F}_0$ the factorizable submodel of $\mathcal{M}$ since $p(s) \in \mathcal{F}_0$ can be factorized with respect to $s_i$. On $\mathcal{F}_0$ the problem is easy: Since $w^I$ are all zero, $s_i$ are statistically independent of others, and therefore $\eta_i = \tanh^{-1} h^i$ and $\eta_{ij} = \eta_i \eta_j$ hold.

Mean field approximation systematically reduces the problem onto the factorizable submodel $\mathcal{F}_0$. For this reduction, I introduce dual foliations $\mathcal{F}$ and $\mathcal{A}$ onto $\mathcal{M}$. The foliation $\mathcal{F} = \{\mathcal{F}(w)\}$, $\mathcal{M} = \bigcup_w \mathcal{F}(w)$, is parametrized by $w \equiv (w^I)$ and each leaf $\mathcal{F}(w)$ is defined as

$$\mathcal{F}(w) = \{p(s) \mid w^I(p) = w^I\}. \tag{3}$$

The leaf $\mathcal{F}(0)$ is the same as $\mathcal{F}_0$, the factorizable submodel. Each leaf $\mathcal{F}(w)$ is again an exponential family with $h^i$ and $\eta_i$ the canonical and the expectation parameters, respectively. A pair of dual potentials is defined on each leaf, one is the Helmholtz free energy $\tilde{\psi}(p) \equiv \psi(p)$ and another is its Legendre transform, or the Gibbs free energy,

$$\tilde{\phi}(p) \equiv \sum_i h^i(p) \eta_i(p) - \tilde{\psi}(p), \tag{4}$$

and the parameters of $p \in \mathcal{F}(w)$ are given by

$$\eta_i(p) = \partial_i \tilde{\psi}(p), \quad h^i(p) = \partial^i \tilde{\phi}(p), \tag{5}$$

where $\partial_i \equiv (\partial / \partial h^i)$ and $\partial^i \equiv (\partial / \partial \eta_i)$. Another foliation $\mathcal{A} = \{\mathcal{A}(m)\}$, $\mathcal{M} = \bigcup_m \mathcal{A}(m)$, is parametrized by $m \equiv (m_i)$ and each leaf $\mathcal{A}(m)$ is defined as

$$\mathcal{A}(m) = \{p(s) \mid \eta_i(p) = m_i\}. \tag{6}$$

Each leaf $\mathcal{A}(\boldsymbol{m})$ is not an exponential family, but again a pair of dual potentials $\tilde{\psi}$ and $\tilde{\phi}$ is defined on each leaf, the former is given by

$$\bar{\psi}(p) = \psi(p) - \sum_i h^i(p)\eta_i(p) \quad (= -\bar{\phi}(p)) \tag{7}$$

and the latter by its Legendre transform as

$$\bar{\phi}(p) = \sum_I w^I(p)\eta_I(p) - \bar{\psi}(p), \tag{8}$$

and the parameters of $p \in \mathcal{A}(\boldsymbol{m})$ are given by

$$\eta_I(p) = \partial_I \bar{\psi}(p), \quad w^I(p) = \partial^I \bar{\phi}(p), \tag{9}$$

where $\partial_I \equiv (\partial/\partial w^I)$ and $\partial^I \equiv (\partial/\partial \eta_I)$. These two foliations form the orthogonal dual foliations, since the leaves $\mathcal{F}(\boldsymbol{w})$ and $\mathcal{A}(\boldsymbol{m})$ are orthogonal at their intersecting point. I introduce still another coordinate system on $\mathcal{M}$, called the mixed coordinate system, on the basis of the orthogonal dual foliations. It uses a pair $(\boldsymbol{m}, \boldsymbol{w})$ of the expectation and the canonical parameters to specify a single element $p \in \mathcal{M}$. The $\boldsymbol{m}$ part specifies the leaf $\mathcal{A}(\boldsymbol{m})$ on which $p$ resides, and the $\boldsymbol{w}$ part specifies the leaf $\mathcal{F}(\boldsymbol{w})$.

## 3.2 REFORMULATION OF PROBLEM

Assume that a target Boltzmann machine $q$ is given by specifying its parameters $h^i(q)$ and $w^I(q)$. Problem 1 is restated as follows: evaluate its expectations $\eta_i(q)$ and $\eta_I(q)$ from those parameters. To evaluate $\eta_i$ mean field approximation translates the problem into the following one:

**Problem 2**  Let $\mathcal{F}(\boldsymbol{w})$ be a leaf on which $q$ resides. Find $p \in \mathcal{F}(\boldsymbol{w})$ which is the closest to $q$.

At first sight this problem is trivial, since one immediately finds the solution $p = q$. However, solving this problem with respect to $\eta_i(p)$ is nontrivial, and it is the key to understanding of mean field approximation including advanced theories.

Let us measure the proximity of $p$ to $q$ by the Kullback divergence

$$D(p\|q) = \sum_s p(s) \log \frac{p(s)}{q(s)}, \tag{10}$$

then solving Problem 2 reduces to finding a minimizer $p \in \mathcal{F}(\boldsymbol{w})$ of $D(p\|q)$ for a given $q$. For $p, q \in \mathcal{F}(\boldsymbol{w})$, $D(p\|q)$ is expressed in terms of the dual potentials $\bar{\psi}$ and $\bar{\phi}$ of $\mathcal{F}(\boldsymbol{w})$ as

$$D(p\|q) = \bar{\psi}(q) + \bar{\phi}(p) - \sum_i h^i(q)\eta_i(p). \tag{11}$$

The minimization problem is thus equivalent to minimizing

$$G(p) \equiv \bar{\phi}(p) - \sum_i h^i(q)\eta_i(p), \tag{12}$$

since $\bar{\psi}(q)$ in eq. (11) does not depend on $p$. Solving the stationary condition $\partial^i G(p) = 0$ with respect to $\eta_i(p)$ will give the correct expectations $\eta_i(q)$, since the true minimizer is $p = q$. However, the scenario is in general intractable since $\bar{\phi}(p)$ cannot be given explicitly as a function of $\eta_i(p)$.

## 3.3  PLEFKA EXPANSION

The problem is easy if $w^I = 0$. In this case $\bar{\phi}(p)$ is given explicitly as a function of $m_i \equiv \eta_i(p)$ as

$$\bar{\phi}(p) = \frac{1}{2}\sum_i \left[(1+m_i)\log\frac{1+m_i}{2} + (1-m_i)\log\frac{1-m_i}{2}\right]. \tag{13}$$

Minimization of $G(p)$ with respect to $m_i$ gives the solution $m_i = \tanh h^i$ as expected. When $w^I \neq 0$ the expression (13) is no longer exact, but to compensate the error one may use, leaving convergence problem aside, the Taylor expansion of $\bar{\phi}(w) \equiv \bar{\phi}(p)$ with respect to $w = 0$,

$$\begin{aligned}\bar{\phi}(w) &= \bar{\phi}(0) + \sum_I (\partial_I\bar{\phi}(0))w^I + \frac{1}{2}\sum_{IJ}(\partial_I\partial_J\bar{\phi}(0))w^I w^J \\ &+ \frac{1}{6}\sum_{IJK}(\partial_I\partial_J\partial_K\bar{\phi}(0))w^I w^J w^K + \cdots.\end{aligned} \tag{14}$$

This expansion has been called the Plefka expansion[12] in the literature of spin glasses. Note that in considering the expansion one should temporarily assume that $m$ is fixed: One can rely on the solution $m$ evaluated from the stationary condition $\partial G(p) = 0$ only if the expansion does not change the value of $m$.

The coefficients in the expansion can be efficiently computed by fully utilizing the orthogonal dual structure of the foliations. First, we have the following theorem:

**Theorem 1** *The coefficients of the expansion (14) are given by the cumulant tensors of the corresponding orders, defined on $\mathcal{A}(m)$.*

Because $\bar{\phi} = -\bar{\psi}$ holds, one can consider derivatives of $\bar{\psi}$ instead of those of $\bar{\phi}$. The first-order derivatives $\partial_I\bar{\psi}$ are immediately given by the property of the potential of the leaf $\mathcal{A}(m)$ (eq. (9)), yielding

$$\partial_I\bar{\psi}(0) = \eta_I(p_0), \tag{15}$$

where $p_0$ denotes the distribution on $\mathcal{A}(m)$ corresponding to $w = 0$. The coefficients of the lowest-orders, including the first-order one, are given by the following theorem.

**Theorem 2** *The first-, second-, and third-order coefficients of the expansion (14) are given by:*

$$\begin{aligned}\partial_I\bar{\psi}(0) &= \eta_I(p_0) \\ \partial_I\partial_J\bar{\psi}(0) &= \langle(\partial_I\ell)(\partial_J\ell)\rangle_{p_0} \\ \partial_I\partial_J\partial_K\bar{\psi}(0) &= \langle(\partial_I\ell)(\partial_J\ell)(\partial_K\ell)\rangle_{p_0}\end{aligned} \tag{16}$$

*where $\ell \equiv \log p_0$.*

The proofs will be found in [11]. It should be noted that, although these results happen to be the same as the ones which would be obtained by regarding $\mathcal{A}(m)$ as an exponential family, they are not the same in general since actually $\mathcal{A}(m)$ is not an exponential family; for example, they are different for the fourth-order coefficients.

The explicit formulas for these coefficients for Boltzmann machines are given as follows:

- For the first-order,

$$\partial_I\bar{\psi}(0) = m_i m_{i'} \quad (I = ii'). \tag{17}$$

- For the second-order,

$$(\partial_I)^2 \bar\psi(\mathbf{0}) = (1 - m_i^2)(1 - m_{i'}^2) \quad (I = ii'),\tag{18}$$

and

$$\partial_I \partial_J \bar\psi(\mathbf{0}) = 0 \quad (I \neq J).\tag{19}$$

- For the third-order,

$$(\partial_I)^3 \bar\psi(\mathbf{0}) = 4 m_i m_{i'} (1 - m_i^2)(1 - m_{i'}^2) \quad (I = ii'),\tag{20}$$

and for $I = ij$, $J = jk$, $K = ik$ for three distinct indices $i$, $j$, and $k$,

$$\partial_I \partial_J \partial_K \bar\psi(\mathbf{0}) = (1 - m_i^2)(1 - m_j^2)(1 - m_k^2)\tag{21}$$

For other combinations of $I$, $J$, and $K$,

$$\partial_I \partial_J \partial_K \bar\psi(\mathbf{0}) = 0.\tag{22}$$

## 4   MEAN FIELD APPROXIMATION

### 4.1   MEAN FIELD EQUATION

Truncating the Plefka expansion (14) up to $n$-th order term gives $n$-th order approximations, $\bar\phi_n(p)$ and $G_n(p) \equiv \bar\phi_n(p) - \sum_i h^i(q) m_i$. The Weiss free energy, which is used in the naive mean field approximation, is given by $\bar\phi_1(p)$. The TAP approach picks up all relevant terms of the Plefka expansion[12], and for the SK model it gives the second-order approximation $\bar\phi_2(p)$.

The stationary condition $\partial^i G_n(p) = 0$ gives the so-called mean field equation, from which a solution of the approximate minimization problem is to be determined. For $n = 1$ it takes the following familiar form,

$$\tanh^{-1} m_i - h^i - \sum_{j \neq i} w^{ij} m_j = 0\tag{23}$$

and for $n = 2$ it includes the so-called Onsager reaction term.

$$\tanh^{-1} m_i - h^i - \sum_{j \neq i} w^{ij} m_j + \sum_{j \neq i} (w^{ij})^2 (1 - m_j^2) m_i = 0\tag{24}$$

Note that all of these are expressed as functions of $m_i$.

Geometrically, the mean field equation approximately represents the "surface" $h^i(p) = h^i(q)$ in terms of the mixed coordinate system of $\mathcal{M}$, since for the exact Gibbs free energy $G$, the stationary condition $\partial^i G(p) = 0$ gives $h^i(p) - h^i(q) = 0$. Accordingly, the approximate relation $h^i(p) = \partial^i \bar\phi_n(p)$, for fixed $\mathbf{m}$, represents the $n$-th order approximate expression of the leaf $\mathcal{A}(\mathbf{m})$ in the canonical coordinate system. The fit of this expression to the true leaf $\mathcal{A}(\mathbf{m})$ around the point $\mathbf{w} = \mathbf{0}$ becomes better as the order of approximation gets higher, as seen in Fig. 1. Such a behavior is well expected, since the Plefka expansion is essentially a Taylor expansion.

### 4.2   LINEAR RESPONSE

For estimating $\eta_I(p)$ one can utilize the linear response theorem. In information geometrical framework it is represented as a trivial identity relation for the Fisher information on the leaf $\mathcal{F}(\mathbf{w})$. The Fisher information matrix $(g_{ij})$, or the Riemannian metric tensor, on the leaf $\mathcal{F}(\mathbf{w})$, and its inverse $(g^{ij})$ are given by

$$g_{ij} = \partial_i \partial_j \bar\psi(p) = \eta_{ij}(p) - \eta_i(p) \eta_j(p)\tag{25}$$

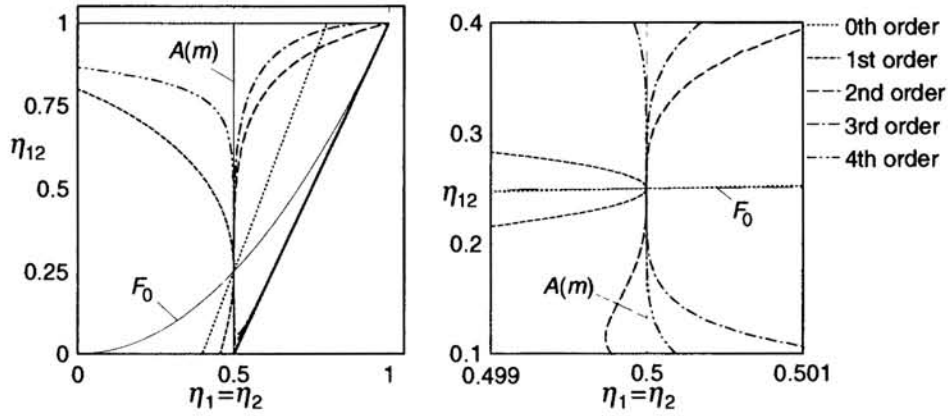

Figure 1: Approximate expressions of $\mathcal{A}(m)$ by mean field approximations of several orders for 2-unit Boltzmann machine, with $(m_1, m_2) = (0.5, 0.5)$ (left), and their magnified view (right).

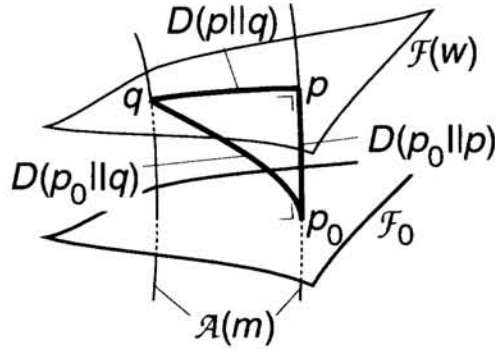

Figure 2: Relation between "naive" approximation and present theory.

and

$$g^{ij} = \partial^i \partial^j \bar{\phi}(p), \tag{26}$$

respectively. In the framework here, the linear response theorem states the trivial fact that those are the inverse of the other. In mean field approximation, one substitutes an approximation $\bar{\phi}_n(p)$ in place of $\bar{\phi}(p)$ in eq. (26) to get an approximate inverse of the metric $(g_n^{ij})$. The derivatives in eq. (26) can be analytically calculated, and therefore $(g_n^{ij})$ can be numerically evaluated by substituting to it a solution $m_i$ of the mean field equation. Equating its inverse to $(g_{ij})$ gives an estimate of $\eta_{ij}(p)$ by using eq. (25). So far, Problem 1 has been solved within the framework of mean field approximation, with $m_i$ and $\eta_{ij}$ obtained by the mean field equation and the linear response theorem, respectively.

## 5   DISCUSSION

Following the framework presented so far, one can in principle construct algorithms of mean field approximation of desired orders. The first-order algorithm with linear response has been first proposed and examined by Kappen and Rodríguez[7, 8]. Tanaka[13] has formulated second- and third-order algorithms and explored them by computer simulations. It is also possible to extend the present formulation so that it can be applicable to higher-order Boltzmann machines. Tanaka[14] discusses an extension of the present formulation to third-order Boltzmann machines: It is possible to extend linear response theorem to higher-orders, and it allows us to treat higher-order correlations within the framework of mean field approximation.

The common understanding about the "naive" mean field approximation is that it minimizes Kullback divergence $D(p_0\|q)$ with respect to $p_0 \in \mathcal{F}_0$ for a given $q$. It can be shown that this view is consistent with the theory presented in this paper. Assume that $q \in \mathcal{F}(\boldsymbol{w})$ and $p_0 \in \mathcal{A}(\boldsymbol{m})$, and let $p$ be a distribution corresponding the intersecting point of the leaves $\mathcal{F}(\boldsymbol{w})$ and $\mathcal{A}(\boldsymbol{m})$. Because of the orthogonality of the two foliations $\mathcal{F}$ and $\mathcal{A}$ the following "Pythagorean law[9]" holds (Fig. 2).

$$D(p_0\|q) = D(p_0\|p) + D(p\|q) \tag{27}$$

Intuitively, $D(p_0\|p)$ measures the squared distance between $\mathcal{F}(\boldsymbol{w})$ and $\mathcal{F}_0$, and is a second-order quantity in $\boldsymbol{w}$. It should be ignored in the first-order approximation, and thus $D(p_0\|q) \approx D(p\|q)$ holds. Under this approximation minimization of the former with respect to $p_0$ is equivalent to that of the latter with respect to $p$, which establishes the relation between the "naive" approximation and the present theory. It can also be checked directly that the first-order approximation of $D(p\|q)$ exactly gives $D(p_0\|q)$, the Weiss free energy.

The present theory provides an alternative view about the validity of mean field approximation: As opposed to a common "belief" that mean field approximation is a good one when $N$ is sufficiently large, one can state from the present formulation that it is so whenever higher-order contribution of the Plefka expansion vanishes, *regardless of whether N is large or not*. This provides a theoretical basis for the observation that mean field approximation often works well for small networks.

The author would like to thank the Telecommunications Advancement Foundation for financial support.

## References

[1] Ackley, D. H., Hinton, G. E., and Sejnowski, T. J. (1985) A learning algorithm for Boltzmann machines. *Cognitive Science* **9**: 147–169.

[2] Peterson, C., and Anderson, J. R. (1987) A mean field theory learning algorithm for neural networks. *Complex Systems* **1**: 995–1019.

[3] Thouless, D. J., Anderson, P. W., and Palmer, R. G. (1977) Solution of 'Solvable model of a spin glass'. *Phil. Mag.* **35** (3): 593–601.

[4] Parisi, G. (1988) *Statistical Field Theory*. Addison-Wesley.

[5] Galland, C. C. (1993) The limitations of deterministic Boltzmann machine learning. *Network* **4** (3): 355–379.

[6] Hofmann, T. and Buhmann, J. M. (1997) Pairwise data clustering by deterministic annealing. *IEEE Trans. Patt. Anal. & Machine Intell.* **19** (1): 1–14; Errata, *ibid.* **19** (2): 197 (1997).

[7] Kappen, H. J. and Rodríguez, F. B. (1998) Efficient learning in Boltzmann machines using linear response theory. *Neural Computation.* **10** (5): 1137–1156.

[8] Kappen, H. J. and Rodríguez, F. B. (1998) Boltzmann machine learning using mean field theory and linear response correction. In M. I. Jordan, M. J. Kearns, and S. A. Solla (Eds.), *Advances in Neural Information Processing Systems* 10, pp. 280–286. The MIT Press.

[9] Amari, S.-I. (1985) *Differential-Geometrical Method in Statistics*. Lecture Notes in Statistics **28**, Springer-Verlag.

[10] Amari, S.-I., Kurata, K., and Nagaoka, H. (1992) Information geometry of Boltzmann machines. *IEEE Trans. Neural Networks* **3** (2): 260–271.

[11] Tanaka, T. Information geometry of mean field approximation. preprint.

[12] Plefka, P. (1982) Convergence condition of the TAP equation for the infinite-ranged Ising spin glass model. *J. Phys. A: Math. Gen.* **15** (6): 1971–1978.

[13] Tanaka, T. (1998) Mean field theory of Boltzmann machine learning. *Phys. Rev. E.* **58** (2): 2302–2310.

[14] Tanaka, T. (1998) Estimation of third-order correlations within mean field approximation. In S. Usui and T. Omori (Eds.), *Proc. Fifth International Conference on Neural Information Processing*, vol. 1, pp. 554–557.
